# Optimal Neural Tuning Curves for Arbitrary Stimulus Distributions: Discrimax, Infomax and Minimum $L_p$ Loss

**Zhuo Wang**
Department of Mathematics
University of Pennsylvania
Philadelphia, PA 19104
wangzhuo@sas.upenn.edu

**Alan A. Stocker**
Department of Psychology
University of Pennsylvania
Philadelphia, PA 19104
astocker@sas.upenn.edu

**Daniel D. Lee**
Department of Electrical and Systems Engineering
University of Pennsylvania
Philadelphia, PA 19104
ddlee@seas.upenn.edu

## Abstract

In this work we study how the stimulus distribution influences the optimal coding of an individual neuron. Closed-form solutions to the optimal sigmoidal tuning curve are provided for a neuron obeying Poisson statistics under a given stimulus distribution. We consider a variety of optimality criteria, including maximizing discriminability, maximizing mutual information and minimizing estimation error under a general $L_p$ norm. We generalize the Cramer-Rao lower bound and show how the $L_p$ loss can be written as a functional of the Fisher Information in the asymptotic limit, by proving the moment convergence of certain functions of Poisson random variables. In this manner, we show how the optimal tuning curve depends upon the loss function, and the equivalence of maximizing mutual information with minimizing $L_p$ loss in the limit as $p$ goes to zero.

## 1 Introduction

A neuron represents sensory information via its spike train. Rate coding maps an input stimulus to a spiking rate via the neuron's tuning. Previous work in computational neuroscience has tried to explain this mapping via optimality criteria. An important factor determining the optimal shape of the tuning curve is the input statistics of the stimulus. It has previously been observed that environmental statistics can influence the neural tuning curves of sensory neurons [1, 2, 3, 4, 5]. However, most theoretical analysis has usually assumed the input stimulus distribution to be uniform. Only recently, theoretical work has been demonstrating how non-uniform prior distributions will affect the optimal shape of the neural tuning curves [6, 7, 8, 9, 10].

An important factor in determining the optimal tuning curve is the optimality criterion [11]. Most previous work used local Fisher Information [12, 13, 14], the estimation square loss or discriminability (*discrimax*) [15, 16] or the mutual information (*infomax*) [9, 17] to evaluate neural codes. It has been shown that both the square loss and the mutual information are related to the Fisher Information via lower bounds: the lower bound of estimation square loss is provided by the Cramer-Rao lower bound [18, 19] and the mutual information can be lower bounded by a functional of Fisher Information as well [7]. It has also been proved that both lower bounds can be attained on the con-

dition that the encoding time is long enough and the estimator behaves well in the asymptotic limit. However, there has been no previous study to integrate those two lower bounds into a more general framework.

In this paper, we ask the question what tuning curve is optimally encoding a stimulus with an arbitrary prior distribution such that the $L_p$ estimation lost is minimized. We are able to provide analytical solutions to the above question. With the asymptotic analysis of the maximum likelihood estimator (MLE), we can show how the $L_p$ loss converges to a functional of Fisher Information in the limit of long encoding time. The optimization of such functional can be conducted for arbitrary stimulus prior and for all $p \geq 0$ in general. The special case of $p = 2$ and the limit $p \rightarrow 0$ corresponds to *discrimax* and *infomax*, respectively. The general result offers a framework to help us understand the *infomax* problem in a new point of view: maximizing mutual information is equivalent to minimizing the expected $L_0$ loss.

## 2 Model and Methods

### 2.1 Encoding and Decoding Model

Throughout this paper we denote $s$ as the scalar stimulus. The stimulus follows an arbitrary prior distribution $\pi(s)$. The encoding process involves a probabilistic mapping from stimulus to a random number of spikes. For each $s$, the neuron will fire at a predetermined firing rate $h(s)$, representing the neuron's tuning curve. The encoded information will contain some noise due to neural variability. According to the conventional Poisson noise model, if the available coding time is $T$, then the observed spike count $N$ has a Poisson distribution with parameter $\lambda = h(s)T$

$$\mathbf{P}[N = k] = \frac{1}{k!} \left( h(s)T \right)^k e^{-h(s)T} \tag{1}$$

The tuning curve $h(s)$ is assumed to be sigmoidal, i.e. monotonically increasing, but limited to a certain range $h_{\min} \leq h(s) \leq h_{\max}$ due to biological constraints. The decoding process is the reverse process of encoding. The estimator $\hat{s} = \hat{s}(N)$ should be a function of observed count $N$. One conventional choice is to use the MLE estimator. First the MLE estimator $\hat{\lambda}$ for mean firing rate is $\hat{\lambda} = N/T$. There for the MLE estimator for stimulus $s$ is simply $\hat{s} = h^{-1}(\hat{\lambda})$.

### 2.2 Fisher Information and Reversal Formula

The Fisher Information can be used to describe how well one can distinguish a specific distribution from its neighboring distributions within the same family of distributions. For a family of distribution with scalar parameter $s$, the Fisher Information is defined as

$$\mathcal{I}(s) = \int \left( \frac{\partial}{\partial s} \log \mathbf{P}(N|s) \right)^2 \mathbf{P}(N|s) \, dN. \tag{2}$$

For tuning function $h(s)$ with Poisson spiking model, the Fisher Information is (see [12, 7])

$$\mathcal{I}_h(s) = T \frac{h'(s)^2}{h(s)} \tag{3}$$

Further with the sigmoidal assumption, by solving the above ordinary differential equation, we can derive the inverse formula in Eq.(4) and an equivalent constraint on Fisher Information in Eq.(5)

$$h(s) = \left( \sqrt{h_{\min}} + \frac{1}{2\sqrt{T}} \int_{-\infty}^{s} \sqrt{\mathcal{I}_h(t)} \, dt \right)^2 \tag{4}$$

$$\int_{-\infty}^{s} \sqrt{\mathcal{I}_h(t)} \, dt \leq 2\sqrt{T} \left( \sqrt{h_{\max}} - \sqrt{h_{\min}} \right) \tag{5}$$

This constraint is closely related to the Jeffrey's prior, which claims that $\pi^*(s) \propto \sqrt{\mathcal{I}(s)}$ is the least informative prior. The above inequality means that the normalization factor of the Jeffrey's prior is finite, as long as the range of firing rate is limited $h_{\min} \leq h(s) \leq h_{\max}$.

## 3 Two Bounds on Loss Function via Fisher Information

### 3.1 Cramer-Rao Bound

The Cramer-Rao Bound [18] for unbiased estimators is

$$\mathbf{E}[(\hat{s} - s)^2 | s] \geq \frac{1}{\mathcal{I}(s)} \tag{6}$$

We can achieve maximum discriminability $\delta^{-1}$ by minimizing the mean asymptotic squared error (MASE), defined in [15] as

$$\delta^2 = \mathbf{E}[(\hat{s} - s)^2] \geq \int \frac{\pi(s)}{\mathcal{I}_h(s)} \, ds, \tag{7}$$

Even if Eq.(7) is only a lower bound, it is attained by the MLE of $s$ asymptotically. In order to optimize the right side of Eq.(7) under the constraints Eq.(5), variation method can be applied and the optimal condition and the optimal solution can be written as

$$\mathcal{I}_h(s) \propto \pi(s)^{2/3}, \quad h_2(s) = \left( \sqrt{h_{\min}} + \left( \sqrt{h_{\max}} - \sqrt{h_{\min}} \right) \frac{\int_{-\infty}^{s} \pi(t)^{1/3} \, dt}{\int_{-\infty}^{\infty} \pi(t)^{1/3} \, dt} \right)^2 \tag{8}$$

### 3.2 Mutual Information Bound

Similar as the Cramer-Rao Bound, Brunel and Nadal [7] gave an upper bound of the mutual information between the MLE $\hat{s}$ and the environmental stimulus $s$

$$I_{\text{mutual}}(\hat{s}, s) \geq H_\pi - \frac{1}{2} \int \pi(s) \log \left( \frac{2\pi e}{\mathcal{I}_h(s)} \right) \, ds, \tag{9}$$

where $H_\pi$ is the entropy of the stimulus prior $\pi(s)$. Although this is an lower bound on the mutual information which we want to maximize, the equality holds asymptotically by the MLE of $s$ as stated in [7]. To maximize the mutual information, we can maximize the right side of Eq.(9) under the constraint of Eq.(5) by variation method again and obtain the optimal condition and optimal solution as

$$\mathcal{I}_h(s) \propto \pi^2(s), \quad h_0(s) = \left( \sqrt{h_{\min}} + \left( \sqrt{h_{\max}} - \sqrt{h_{\min}} \right) \frac{\int_{-\infty}^{s} \pi(t) \, dt}{\int_{-\infty}^{\infty} \pi(t) \, dt} \right)^2 \tag{10}$$

To see the connection between solutions in Eq.(8) and Eq.(10), we need the result of the following section.

## 4 Asymptotic Behavior of Estimators

In general, minimizing the lower bound does not imply that the measures of interest, e.g. the left side of Eq.(7) and Eq.(9), is minimized. In order to make the lower bounds useful, we need to know the conditions for which there exist "good" estimators that can reach these theoretical lower bounds.

First we will introduce some definitions of estimator properties. Let $T$ be the encoding time for a neuron with Poisson noise, and $\hat{s}_T$ be the MLE at time $T$. If we denote $Y'_T = \sqrt{T}(\hat{s}_T - s)$ and $Z' \sim \mathcal{N}(0, T/\mathcal{I}(s))$, then the notions we have mentioned above are defined as below

$$\mathbf{E}[Y'_T] \to 0 \qquad \text{(asymptotic consistency)} \tag{11}$$

$$\mathbf{Var}[Y'_T] \to T/\mathcal{I}(s) \qquad \text{(asymptotic efficiency)} \tag{12}$$

$$Y'_T \xrightarrow{D} Z' \qquad \text{(asymptotic normality)} \tag{13}$$

$$\mathbf{E}[|Y'_T|^p] \to \mathbf{E}[|Z'|^p] \qquad \text{($p$-th moment convergence)} \tag{14}$$

Generally the above four conditions are listed from the weakest to the strongest, top to bottom. To have the equality in Eq.(7) hold, we need the *asymptotic consistent* and *asymptotic efficient* estimators. To have the equality in (9) hold, we need the *asymptotic normal* estimators (see [7]). If

we want to generalize the problem even further, i.e. finding the tuning curve which minimizes the $L_p$ estimation loss, then we need the *moment convergent* estimator for all $p$-th moments.

Here we will give two theorems to prove that the MLE $\hat{s}$ of the true stimulus $s$ would satisfy all the above four properties in Eq.(11)-(14). Let $h(s)$ be the tuning curve of a neuron with Poisson spiking noise. The the MLE of $s$ is given by $\hat{s} = h^{-1}(\hat{\lambda})$. We will show that the limiting distribution of $\sqrt{T}(\hat{s}_T - s)$ is a Gaussian distribution with mean 0 and variance $h(s)/h'(s)^2$. We will also show that any positive $p$-norm of $\sqrt{T}(\hat{s}_T - s)$ will converge the $p$-norm of the corresponding Gaussian distribution. The proof of Theorem 1 and 2 will be provided in Appendix A.

**Theorem 1.** *Let $X_i$ be i.i.d. Poisson distributed random variables with mean $\lambda$. Let $S_n = \sum_{i=1}^{n} X_i$ be the partial sum. Then*

(a) *$S_n$ has Poisson distribution with mean $n\lambda$.*

(b) *$Y_n = \sqrt{n}(S_n/n - \lambda)$ converges to $Z \sim \mathcal{N}(0, \lambda)$ in distribution.*

(c) *The $p$-th moment of $Y_n$ converges, and $\lim_{n\to\infty} \mathbf{E}_\lambda[|Y_n|^p] = \mathbf{E}[|Z|^p]$ for all $p > 0$.*

One direct application of this theorem is that, if the tuning curve $h(s) = s$ for $(s > 0)$ and the encoding time is $T$, then the estimator $\hat{s} = N/T$ is asymptotically efficient since as $T \to \infty$, $\mathbf{Var}[\hat{s}] \to \mathbf{E}[|Z_\lambda/\sqrt{T}|^2] = s/T = 1/\mathcal{I}(s)$.

**Theorem 2.** *Let $X_i, S_n$ be defined as in Theorem 1. Let $g(x)$ be any function with bounded derivative $|g'(x)| \le M$. Then*

(a) *$Y_n' = \sqrt{n}(g(S_n/n) - g(\lambda))$ converges to $Z' \sim \mathcal{N}(0, \lambda g'(\lambda)^2)$ in distribution.*

(b) *The $p$-th moment of $Y_n'$ converges, and $\lim_{n\to\infty} \mathbf{E}_\lambda[|Y_n'|^p] = \mathbf{E}[|Z'|^p]$ for all $p > 0$.*

Theorem 1 indicates that we can always estimate the firing rate $\lambda = h(s)$ efficiently by the estimator $\hat{\lambda} = N/T$. Theorem 2 indicates, however, that we can also estimate a smooth transformation of the firing rate efficiently in the asymptotic limit $T \to \infty$. Now, if we go back to the conventional setting of the tuning curve $\lambda = h(s)$, we can estimate the stimulus by the estimator $\hat{s} = h^{-1}(\hat{\lambda})$. To meet the need of boundedness of $g$: $|g'(\lambda)| \le M$, we have $1/g'(\lambda) = h'(s) \ge 1/M$ hence this theory only works for stimulus from a compact set $s \in [-M, M]$, although the $M$ can be chosen as large as possible. The larger the $M$ is, the longer encoding time $T$ will be necessary to observe the asymptotic normality and the convergence of moments.

The estimator $\hat{s} = h^{-1}(\hat{\lambda})$ is biased for finite $T$, but it is asymptotically unbiased and efficient. This is because as $T \to \infty$

$$\mathbf{E}_s[\sqrt{T}(\hat{s}_T - s)] \to \mathbf{E}[Z'] = 0 \tag{15}$$

$$\mathbf{Var}_s[\sqrt{T}(\hat{s}_T - s)] \to \mathbf{E}[|Z'|^2] = \lambda(h^{-1})'(\lambda)^2 = \frac{h(s)}{h'(s)^2} = \frac{T}{\mathcal{I}(s)} \tag{16}$$

From the above analysis we can see that the property of $L_p(\hat{s}, s) = \mathbf{E}_s[|\hat{s}_T - s|^p]$ saturating the lower bound fully relies upon the asymptotic normality. With asymptotic normality, we can do more than just optimizing $I_{\text{mutual}}(N, s)$ and $L_p(\hat{s}, s)$. In general we can find the optimal tuning curve which minimizes the expected $L_p$ loss $L_p(\hat{s}, s)$ since as $T \to \infty$

$$\mathbf{E}\left[\left|\sqrt{T}(\hat{s}_T - s)\right|^p\right] \to \mathbf{E}\left[|Z'|^p\right] \tag{17}$$

where $Z' = \chi/\sqrt{\mathcal{I}(s)/T}$, $\chi \sim \mathcal{N}(0, 1)$. To calculate the right side of the above limit, we can use the fact that for any $p \ge 0$,

$$K(p) = \mathbf{E}\left[|\chi|^p\right] = \left(\sqrt{2}\right)^p \frac{\Gamma\left(\frac{p+1}{2}\right)}{\Gamma\left(\frac{1}{2}\right)} \tag{18}$$

where $\Gamma(\cdot)$ is the gamma function

$$\Gamma(z) = \int_0^\infty e^{-t} t^{z-1} \, dt \tag{19}$$

The general conclusion is that (Cramer-Rao Lower bound is a special case with $p = 2$)

$$\mathbf{E}_s\left[\left|\sqrt{T}(\hat{s}_T - s)\right|^p\right] \to \mathbf{E}\left[|Z'|^p\right] = \frac{K(p)}{(\mathcal{I}(s)/T)^{p/2}} \tag{20}$$

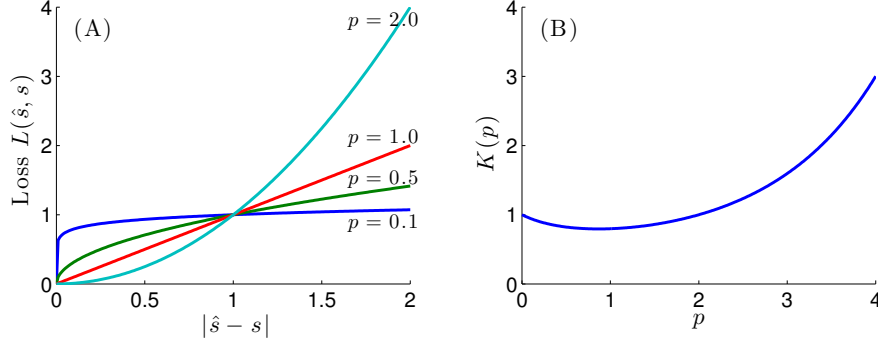

Figure 1: (A) Illustration of $L_p$-loss as a function of $|\hat{s} - s|$ for different values of $p$. When $p = 2$ the loss is the squared loss and when $p \to 0$, the $L_p$ loss converges to 0-1 loss pointwise. (B) The plot of $p$-th absolute moments $K(p) = \mathbf{E}[|\chi|^p]$ of standard Gaussian random variable $\chi$ for $p \in [0, 4]$.

## 5    Optimal Tuning Curves: Infomax, Discrimax and More

With the asymptotic normality and moment convergence, we know the asymptotic expected $L_p$ loss will approach $\mathbf{E}[|Z'|^p]$ for each $s$. Hence

$$\mathbf{E}\left[|\hat{s} - s|^p\right] \to \int \pi(s)\mathbf{E}_s\left[|Z'|^p\right]ds = K(p)\int \frac{\pi(s)}{\mathcal{I}(s)^{p/2}}ds. \tag{21}$$

To obtain the optimal tuning curve for the $L_p$ loss, we need to solve a simple variation problem

$$\underset{h}{\text{minimize}} \quad \int \pi(s)f(\mathcal{I}_h(s))\,ds \tag{22}$$

$$\text{subject to} \int \sqrt{\mathcal{I}_h(s)}\,ds \leq const \tag{23}$$

with $f_p'(x) = -x^{-p/2-1}$. To solve this variational problem, the Euler-Lagrange equation and the Lagrange multiplier method can be used to derive the optimal condition

$$0 = \frac{\partial}{\partial \mathcal{I}_h}\left(\pi(s)f_p(\mathcal{I}_h(s)) - \lambda\sqrt{\mathcal{I}_h}\right) = \pi(s)f_p'\left(\mathcal{I}_h(s)\right) - \frac{\lambda}{2}\mathcal{I}_h(s)^{-1/2} \tag{24}$$

$$\Rightarrow \quad \sqrt{\mathcal{I}_h(s)} \propto \pi(s)^{1/(p+1)} \tag{25}$$

Therefore the $f_p$-optimal tuning curve, which minimizes the asymptotic $L_p$ loss, is given by equation below, followed from (4) and (25). For some examples of $L_p$ optimal tuning curves, see Fig. 2.

$$h_p(s) = \left(\sqrt{h_{\min}} + \left(\sqrt{h_{\max}} - \sqrt{h_{\min}}\right)\frac{\int_{-\infty}^{s}\pi(t)^{1/(p+1)}\,dt}{\int_{-\infty}^{\infty}\pi(t)^{1/(p+1)}\,dt}\right)^2 \tag{26}$$

$$\mathcal{I}_p(s) = 4T\left(\sqrt{h_{\max}} - \sqrt{h_{\min}}\right)^2 \frac{\pi(s)^{2/(p+1)}}{\left(\int \pi(t)^{1/(p+1)}\,dt\right)^2} \tag{27}$$

Following from (21) and (27), the optimal expected $L_p$ loss is

$$\mathbf{E}\left[|\hat{s} - s|^p\right] = K(p)\cdot(4T)^{-p/2}\left(\sqrt{h_{\max}} - \sqrt{h_{\min}}\right)^{-p}\left(\int \pi(t)^{1/(p+1)}\,dt\right)^{p+1} \tag{28}$$

A very interesting observation is that, by taking the limit $p \to 0$, we will end up with the *infomax* tuning curve. This shows that the *infomax* tuning curve simultaneously optimizes the mutual information as well as the expected $L_0$ norm of the error $\hat{s} - s$. The $L_0$ norm can be considered as the 0-1 loss, i.e. $L(\hat{s}, s) = 0$ if $\hat{s} = s$ and $L(\hat{s}, s) = 1$ otherwise. To put this in a different approach, we may consider the natural log function as a limit of power function:

$$\log z = \lim_{p \to 0} \frac{1 - z^{-p/2}}{p/2} \tag{29}$$

$$\Rightarrow \int \pi(s) \log \mathcal{I}(s)\, ds = \lim_{p \to 0} \frac{2}{p} \left( 1 - \int \pi(s) \mathcal{I}(s)^{-p/2}\, ds \right) \tag{30}$$

and we can conclude that minimizing $\int \pi(s) \mathcal{I}(s)^{-p/2} ds$ in the limit of $p \to 0$ ($L_0$ loss) is the same as maximizing $\int \pi(s) \log \mathcal{I}(s) ds$ and the mutual information.

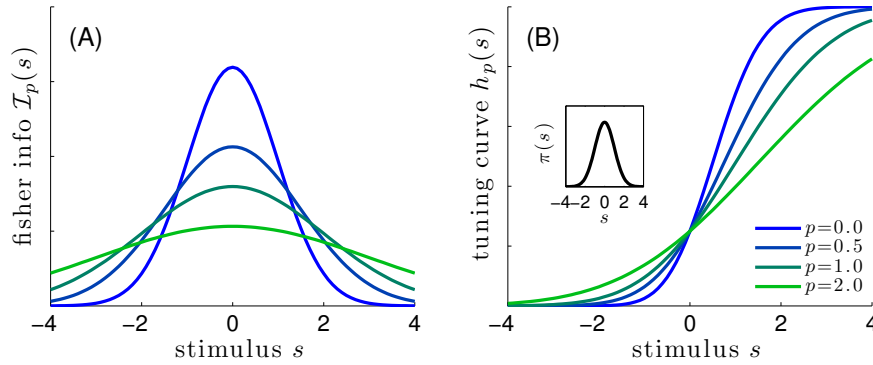

Figure 2: For stimulus with standard Gaussian prior distribution (inset figure) and various values of $p$, (A) shows the optimal allocation of Fisher Information $\mathcal{I}_p(s)$ and (B) shows the $f_p$-optimal tuning curve $h_p(s)$. When $p = 2$ the $f_2$-optimal (*discrimax*) tuning curve minimizes the squared loss and when $p = 0$ the $f_0$-optimal (*infomax*) tuning curve maximizes the mutual information.

## 6 Simulation Results

Numerical simulations were performed in order to validate our theory. In each iteration, a random stimulus $s$ was chosen from the standard Gaussian distribution or Exponential distribution with mean one. A Poisson neuron was simulated to generate spikes in response to that stimulus. The difference between the MLE $\hat{s}$ and $s$ is recorded to analyze the $L_p$ loss. In one simple task, we compared the numerical value vs. the theoretical value of $L_p$ loss for $f_q$-optimal tuning curve

$$\mathbf{E}\left[|\hat{s} - s|^p\right] = K(p) \cdot (4T)^{-p/2} \left( \sqrt{h_{\max}} - \sqrt{h_{\min}} \right)^{-p} \left( \int \pi(t)^{1/(q+1)}\, dt \right)^p \left( \int \pi(s)^{1 - \frac{p}{q+1}}\, ds \right) \tag{31}$$

The above theoretical prediction works well for distributions with compact support $s \in [A, B]$. It also requires $q > p - 1$ for any distribution with tail decaying faster than exponential: $\pi(s) \le e^{-Cs}$, such as *e.g.* a Gaussian or exponential distribution. Otherwise the integral in the last term will blow up in general.

The numerical and theoretical prediction of $L_p$ loss are plotted, for both Gaussian $\mathcal{N}(0, 1)$ prior (Fig.3A) and Exp(1) prior (Fig.3B). The vertical axis shows $1/p \cdot \log \mathbf{E}[|\hat{s} - s|^p]$ so all $p$-norms are displayed at the same unit.

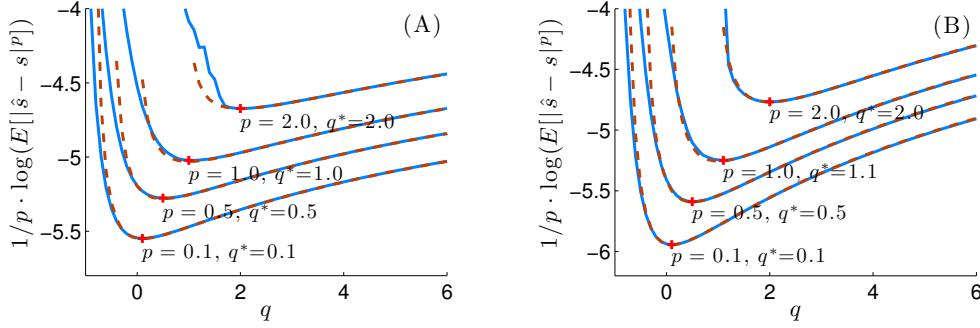

Figure 3: The comparison between numerical result (solid curves) and theoretical prediction (dashed curves). (A) For standard Gaussian prior. (B) For exponential prior with parameter 1.

## 7 Discussion

In this paper we have derived a closed form solution for the optimal tuning curve of a single neuron given an arbitrary stimulus prior $\pi(s)$ and for a variety of optimality. Our work offers a principled explanation for the observed non-linearity in neuron tuning: Each neuron should adapt their tuning curves to reallocate the limited amount of Fisher information they can carry and minimize the $L_p$ error. We have shown in section 2 that each sigmoidal tuned neuron with Poisson spiking noise has an upper bound for the integral of square root of Fisher information and the $f_p$-optimal tuning curve has the form

$$h_p(s) = \left( \sqrt{h_{\min}} + \left( \sqrt{h_{\max}} - \sqrt{h_{\min}} \right) \frac{\int_{-\infty}^{s} \pi(t)^{1/(p+1)}\, dt}{\int_{-\infty}^{\infty} \pi(t)^{1/(p+1)}\, dt} \right)^2 \tag{32}$$

where the $f_p$-optimal tuning curve minimizes the estimation $L_p$ loss $\mathbf{E}[|\hat{s} - s|^p]$ of the decoding process in the limit of long encoding time $T$. Two special and well known cases are maximum mutual information ($p = 0$) and maximum discriminant ($p = 2$).

To obtain this result, we established two theorems regarding the asymptotic behavior of the MLE $\hat{s} = h^{-1}(\hat{\lambda})$. Asymptotically, the MLE converges to a standard Gaussian not only with regard to its distribution, but also in terms of its $p$-th moment for arbitrary $p > 0$. By calculating the $p$-th moments for the Gaussian random variable, we can predict the $L_p$ loss of the encoding-decoding process and optimize the tuning curve by minimizing the attainable limit. The Cramer-Rao lower bound and the mutual information lower bound proposed by [7] are special cases with $p = 2$ or $p = 0$ respectively.

So far, we have put our focus on a single neuron with sigmoidal tuning curve. However, the conclusions in Theorem 1 and 2 still hold for the case of neuronal populations with bell-shaped neurons, with correlated or uncorrelated noise. The optimal condition for Fisher information can be calculated, regardless of the tuning curve(s) format. According to the assumption on the number of neurons and the shape of the tuning curves, the optimized Fisher information can be inverted to derive the optimal tuning curve via the same type of analysis as we presented in this paper.

One theoretic limitation of our method is that we only addressed the problem for long encoding times, which is usually not the typical scenario in real sensory systems. Though the long encoding time limit can be replaced by short encoding time with many identical tuned neurons. It is still an interesting problem to find out the optimal tuning curve for arbitrary prior, in the sense of $L_p$ loss function. Some work [16, 20] has been done to maximize mutual information or $L_2$ for uniformly distributed stimuli. Another problem is that the asymptotic behavior is not uniformly true if the space of stimulus is not compact. The asymptotic behavior will take longer to be observed if the slop of the tuning function is too close to zero. In Theorem 2 we made the assumption that $|g'(s)| \leq M$ and that is the reason we cannot evaluate the estimation error for $s$ with large absolute value hence we do not have a perfect match for low $p$ values in the simulation section (see Fig. 3).

# A  Proof of Theorems in Section 4

*Proof.* of Theorem 1

**(a)** Immediately follows from Poisson distribution. Use induction on $n$.

**(b)** Apply Central Limit Theorem. Notice that $\mathbf{E}[X_i] = \mathbf{Var}[X_i] = \lambda$ for Poisson random variables.

**(c)** In general, convergence in distribution does not imply convergence in $p$-th moment. However in our case, we do have the convergence property for all $p$-th moments. To show this, we need to show for all $p > 0$, $|Y_n|^p$ is uniformly integrable i.e. for any $\epsilon$, there exist a $K$ such that

$$\mathbf{E}[|Y_n|^p \cdot 1_{\{|Y_n| \geq K\}}] \leq \epsilon \tag{33}$$

This is obvious with Cauchy-Schwartz inequality and Markov inequality

$$\left(\mathbf{E}[|Y_n|^p \cdot 1_{\{Y_n \geq K\}}]\right)^2 \leq \mathbf{E}[|Y_n|^{2p}] \cdot \mathbf{P}[|Y_n| \geq K] \leq \mathbf{E}[|Y_n|^{2p}]\frac{E[|Y_n|]}{K} \to 0 \tag{34}$$

To see the last limit, we use the fact that for all $p > 0$, $\sup_n \mathbf{E}[|Y_n|^p] < \infty$. According to [21],

$$\mathbf{E}[|S_n - n\lambda|^p] = \sum_{a=0}^{p}(n\lambda)^a S_2(p, a), \tag{35}$$

where $S_2(p, a)$ denotes the number of partitions of a set of size $n$ into $a$ subsets with no singletons (i.e. no subsets with only one element). For our purpose, notice that $S_2(p, a) = 0$ for $a > p/2$ and $S_2(p, a) \leq p^a$. Therefore the supreme of $\mathbf{E}[|Y_n|^p]$ is bounded since

$$\mathbf{E}[|Y_n|^p] = \mathbf{E}[|\sqrt{n}(S_n/n - \lambda)|^p] \leq n^{-p/2}\sum_{a=0}^{p/2}(n\lambda)^a p^a \leq \frac{n(\lambda p)^{p/2+1}}{n\lambda p - 1} \leq C(\lambda p)^{p/2} \tag{36}$$

For arbitrary $q$, choose any even number $p$ such that $p > q$, and by Jensen's inequality, $\mathbf{E}[|Y_n|^q] \leq \mathbf{E}[|Y_n|^p]^{q/p}$. Thus for all $p > 0, n$, $\mathbf{E}[|Y_n|^p] < \infty$.

$\square$

*Proof.* of Theorem 2

**(a)** Denote $\hat{\lambda}_n = S_n/n$. Apply mean value theorem for $g(x)$ near $\lambda$ :

$$g(\hat{\lambda}_n) - g(\lambda) = g'(\lambda^*)(\hat{\lambda}_n - \lambda) \tag{37}$$

for some $\lambda^*$ between $\hat{\lambda}_n$ and $\lambda$. Therefore

$$\sqrt{n}\left(g(\hat{\lambda}_n) - g(\lambda)\right) = g'(\lambda^*)\sqrt{n}(\hat{\lambda} - \lambda) \xrightarrow{D} g'(\lambda)Z \tag{38}$$

Note that $\hat{\lambda}_n \to \lambda$ in probability, $\lambda^* \to \lambda$ in probability and $g'(\lambda^*) \to g'(\lambda)$ in probability, together with the fact that $\sqrt{n}(\hat{\lambda}_n - \lambda) \xrightarrow{D} Z$, apply Slutsky's theorem and the conclusion follows.

**(b)** Use Taylor's expansion and Slutsky's theorem again,

$$\left|\sqrt{n}\left(g(\hat{\lambda}_n) - g(\lambda)\right)\right|^p = \left|g'(\lambda^*)\sqrt{n}(\hat{\lambda} - \lambda)\right|^p = |g'(\lambda^*)|^p |Y_n|^p \to |g'(\lambda)|^p |Y_n|^p \tag{39}$$

To see $|Y_n'|^p$ is uniformly integrable, notice that $|Y_n'|^p \geq K \Rightarrow |Y_n|^p \geq K \cdot M^{-p}$. The rest follows in a similar manner as when proving Theorem 1(c).

$\square$

# References

[1] TM Maddess and SB Laughlin. Adaptation of the motion-sensitive neuron h1 is generated locally and governed by contrast frequency. *Proc. R. Soc. Lond. B Biol. Sci*, 225:251–275, 1985.

[2] J Atick. Could information theory provide an ecological theory of sensory processing? *Network*, 3:213–251, 1992.

[3] RA Harris, DC O'Carroll, and SB Laughlin. Contrast gain reduction in fly motion adaptation. *Neuron*, 28:595–606, 2000.

[4] I Dean, NS Harper, and D McAlpine. Neural population coding of sound level adapts to stimulus statistics. *Nature neuroscience*, 8:1684–1689, 2005.

[5] AA Stocker and EP Simoncelli. Noise characteristics and prior expectations in human visual speed perception. *Nature neuroscience*, 9:578–585, 2006.

[6] J-P Nadal and N Parga. Non linear neurons in the low noise limit: A factorial code maximizes information transfer, 1994.

[7] N Brunel and J-P Nadal. Mutual information, fisher information and population coding. *Neural Computation*, 10(7):1731–1757, 1998.

[8] Tvd Twer and DIA MacLeod. Optimal nonlinear codes for the perception of natural colours. *Network: Computation in Neural Systems*, 12(3):395–407, 2001.

[9] MD McDonnell and NG Stocks. Maximally informative stimuli and tuning curves for sigmoidal rate-coding neurons and populations. *Phys. Rev. Lett.*, 101:058103, 2008.

[10] D Ganguli and EP Simoncelli. Implicit encoding of prior probabilities in optimal neural populations. *Adv. Neural Information Processing Systems*, 23:658–666, 2010.

[11] HB Barlow. *Possible principles underlying the transformation of sensory messages*. M.I.T. Press, 1961.

[12] HS Seung and H Sompolinsky. Simple models for reading neuronal population codes. *Proc. of the National Aca. of Sci. of the U.S.A.*, 90:10749–10753, 1993.

[13] K Zhang and TJ Sejnowski. Neuronal tuning: To sharpen or broaden? *Neural Computation*, 11:75–84, 1999.

[14] A Pouget, S Deneve, J-C Ducom, and PE Latham. Narrow versus wide tuning curves: Whats best for a population code? *Neural Computation*, 11:85–90, 1999.

[15] M Bethge, D Rotermund, and K Pawelzik. Optimal short-term population coding: when Fisher information fails. *Neural Computation*, 14:2317–2351, 2002.

[16] M Bethge, D Rotermund, and K Pawelzik. Optimal neural rate coding leads to bimodal firing rate distributions. *Netw. Comput. Neural Syst.*, 14:303–319, 2003.

[17] S Yarrow, E Challis, and P Seris. Fisher and shannon information in finite neural populations. *Neural Computation*, In Print, 2012.

[18] TM Cover and J Thomas. *Elements of Information Theory*. Wiley, 1991.

[19] SI Amari, H Nagaoka, and D Harada. *Methods of Information Geometry*. Translations of Mathematical Monographs. American Mathematical Society, 2007.

[20] AP Nikitin, NG Stocks, RP Morse, and MD McDonnell. Neural population coding is optimized by discrete tuning curves. *Phys. Rev. Lett.*, 103:138101, 2009.

[21] N Privault. Generalized Bell polynomials and the combinatorics of Poisson central moments. *Electronic Journal of Combinatorics*, 18, 2011.

